# Dynamics of Supervised Learning with Restricted Training Sets and Noisy Teachers

**A.C.C. Coolen**
Dept of Mathematics
King's College London
The Strand, London WC2R 2LS, UK
*tcoolen@mth.kcl.ac.uk*

**C.W.H. Mace**
Dept of Mathematics
King's College London
The Strand, London WC2R 2LS, UK
*cmace@mth.kcl.ac.uk*

## Abstract

We generalize a recent formalism to describe the dynamics of supervised learning in layered neural networks, in the regime where data recycling is inevitable, to the case of noisy teachers. Our theory generates reliable predictions for the evolution in time of training- and generalization errors, and extends the class of mathematically solvable learning processes in large neural networks to those situations where overfitting can occur.

## 1  Introduction

Tools from statistical mechanics have been used successfully over the last decade to study the dynamics of learning in layered neural networks (for reviews see e.g. [1] or [2]). The simplest theories result upon assuming the data set to be much larger than the number of weight updates made, which rules out recycling and ensures that any distribution of relevance will be Gaussian. Unfortunately, both in terms of applications and in terms of mathematical interest, this regime is not the most relevant one. Most complications and peculiarities in the dynamics of learning arise precisely due to data recycling, which creates for the system the possibility to improve performance by memorizing answers rather than by learning an underlying rule. The dynamics of learning with restricted training sets was first studied analytically in [3] (linear learning rules) and [4] (systems with binary weights). The latter studies were ahead of their time, and did not get the attention they deserved just because at that stage even the simpler learning dynamics without data recycling had not yet been studied. More recently attention has moved back to the dynamics of learning in the recycling regime. Some studies aimed at developing a general theory [5, 6, 7], some at finding exact solutions for special cases [8]. All general theories published so far have in common that they as yet considered realizable scenario's: the rule to be learned was implementable by the student, and overfitting could not yet occur. The next hurdle is that where restricted training sets are combined with unrealizable rules. Again some have turned to non-typical but solvable cases, involving Hebbian rules and noisy [9] or 'reverse wedge' teachers [10]. More recently the cavity method has been used to build a general theory [11] (as yet for batch learning only). In this paper we generalize the general theory launched in [6, 5, 7], which applies to arbitrary learning rules, to the case of noisy teachers. We will mirror closely the presentation in [6] (dealing with the simpler case of noise-free teachers), and we refer to [5, 7] for background reading on the ideas behind the formalism.

## 2  Definitions

As in [6, 5] we restrict ourselves for simplicity to perceptrons. A student perceptron operates a linear separation, parametrised by a weight vector $\boldsymbol{J} \in \Re^N$:

$$S : \{-1,1\}^N \to \{-1,1\} \qquad S(\boldsymbol{\xi}) = \operatorname{sgn}[\boldsymbol{J} \cdot \boldsymbol{\xi}]$$

It aims to emulate a teacher operating a similar rule, which, however, is characterized by a variable weight vector $\boldsymbol{B} \in \Re^N$, drawn at random from a distribution $P(\boldsymbol{B})$ such as

$$\text{output noise}: \qquad P(\boldsymbol{B}) = \lambda\delta[\boldsymbol{B}+\boldsymbol{B}^\star] + (1-\lambda)\delta[\boldsymbol{B}-\boldsymbol{B}^\star] \qquad (1)$$

$$\text{Gaussian weight noise}: \qquad P(\boldsymbol{B}) = [\Sigma\sqrt{2\pi}/N]^{-N}\, e^{-\frac{1}{2}N(\boldsymbol{B}-\boldsymbol{B}^\star)^2/\Sigma^2} \qquad (2)$$

The parameters $\lambda$ and $\Sigma$ control the amount of teacher noise, with the noise-free teacher $\boldsymbol{B} = \boldsymbol{B}^\star$ recovered in the limits $\lambda \to 0$ and $\Sigma \to 0$. The student modifies $\boldsymbol{J}$ iteratively, using examples of input vectors $\boldsymbol{\xi}$ which are drawn at random from a fixed (randomly composed) training set containing $p = \alpha N$ vectors $\boldsymbol{\xi}^\mu \in \{-1,1\}^N$ with $\alpha > 0$, and the corresponding values of the teacher outputs. We choose the teacher noise to be consistent, i.e. the answer given by the teacher to a question $\boldsymbol{\xi}^\mu$ will remain the same when that particular question re-appears during the learning process. Thus $T(\boldsymbol{\xi}^\mu) = \operatorname{sgn}[\boldsymbol{B}^\mu \cdot \boldsymbol{\xi}^\mu]$, with $p$ teacher weight vectors $\boldsymbol{B}^\mu$, drawn randomly and independently from $P(\boldsymbol{B})$, and we generalize the training set accordingly to $\tilde{D} = \{(\boldsymbol{\xi}^1, \boldsymbol{B}^1), \ldots, (\boldsymbol{\xi}^p, \boldsymbol{B}^p)\}$. Consistency of teacher noise is natural in terms of applications, and a prerequisite for overfitting phenomena. Averages over the training set will be denoted as $\langle\ldots\rangle_{\tilde{D}}$; averages over all possible input vectors $\boldsymbol{\xi} \in \{-1,1\}^N$ as $\langle\ldots\rangle_{\boldsymbol{\xi}}$. We analyze two classes of learning rules, of the form $\boldsymbol{J}(\ell+1) = \boldsymbol{J}(\ell) + \Delta\boldsymbol{J}(\ell)$:

$$\text{on}-\text{line}: \quad \Delta\boldsymbol{J}(\ell) = \tfrac{\eta}{N}\{\boldsymbol{\xi}(\ell)\, \mathcal{G}[\boldsymbol{J}(\ell)\cdot\boldsymbol{\xi}(\ell), \boldsymbol{B}(\ell)\cdot\boldsymbol{\xi}(\ell)] - \gamma\boldsymbol{J}(\ell)\}$$

$$\text{batch}: \quad \Delta\boldsymbol{J}(\ell) = \tfrac{\eta}{N}\{\langle\boldsymbol{\xi}\, \mathcal{G}[\boldsymbol{J}(\ell)\cdot\boldsymbol{\xi}, \boldsymbol{B}\cdot\boldsymbol{\xi}]\rangle_{\tilde{D}} - \gamma\boldsymbol{J}(m)\} \qquad (3)$$

In on-line learning one draws at each step $\ell$ a question/answer pair $(\boldsymbol{\xi}(\ell), \boldsymbol{B}(\ell))$ at random from the training set. In batch learning one iterates a deterministic map which is an average over all data in the training set. Our performance measures are the training- and generalization errors, defined as follows (with the step function $\theta[x > 0] = 1$, $\theta[x < 0] = 0$):

$$E_{\text{t}}(\boldsymbol{J}) = \langle\theta[-(\boldsymbol{J}\cdot\boldsymbol{\xi})(\boldsymbol{B}\cdot\boldsymbol{\xi})]\rangle_{\tilde{D}} \qquad E_{\text{g}}(\boldsymbol{J}) = \langle\theta[-(\boldsymbol{J}\cdot\boldsymbol{\xi})(\boldsymbol{B}^\star\cdot\boldsymbol{\xi})]\rangle_{\boldsymbol{\xi}} \qquad (4)$$

We introduce macroscopic observables, taylored to the present problem, generalizing [5, 6]:

$$Q[\boldsymbol{J}] = \boldsymbol{J}^2, \quad R[\boldsymbol{J}] = \boldsymbol{J}\cdot\boldsymbol{B}^\star, \quad P[x,y,z;\boldsymbol{J}] = \langle\delta[x-\boldsymbol{J}\cdot\boldsymbol{\xi}]\delta[y-\boldsymbol{B}^\star\cdot\boldsymbol{\xi}]\delta[z-\boldsymbol{B}\cdot\boldsymbol{\xi}]\rangle_{\tilde{D}} \qquad (5)$$

As in [5, 6] we eliminate technical subtleties by assuming the number of arguments $(x, y, z)$ for which $P[x, y, z; \boldsymbol{J}]$ is evaluated to go to infinity after the limit $N \to \infty$ has been taken.

## 3  Derivation of Macroscopic Laws

Upon generalizing the calculations in [6, 5], one finds for on-line learning:

$$\frac{d}{dt}Q = 2\eta\int dxdydz\, P[x,y,z]\, x\mathcal{G}[x,z] - 2\eta\gamma Q + \eta^2\int dxdydz\, P[x,y,z]\, \mathcal{G}^2[x,z] \qquad (6)$$

$$\frac{d}{dt}R = \eta\int dxdydz\, P[x,y,z]\, y\mathcal{G}[x,z] - \eta\gamma R \qquad (7)$$

$$\frac{\partial}{\partial t}P[x,y,z] = \frac{1}{\alpha}\int dx'\, P[x',y,z]\left\{\delta[x-x'-\eta G[x',z]] - \delta[x-x']\right\}$$

$$-\eta\frac{\partial}{\partial x}\int dx'dy'dz'\int dx'dy'dz'\mathcal{G}[x',z]\mathcal{A}[x,y,z;x',y',z'] + \eta\gamma\frac{\partial}{\partial x}\left\{xP[x,y,z]\right\}$$

$$+\frac{1}{2}\eta^2\int dx'dy'dz'\, P[x',y',z']\mathcal{G}^2[x',z]\frac{\partial^2}{\partial x^2}P[x,y,z] \qquad (8)$$

The complexity of the problem is concentrated in a Green's function:

$$\mathcal{A}[x,y,z;x',y',z'] = \lim_{N\to\infty}$$

$$\langle\langle\langle([1-\delta_{\boldsymbol{\xi}\boldsymbol{\xi}'}]\delta[x-\boldsymbol{J}\cdot\boldsymbol{\xi}]\delta[y-\boldsymbol{B}^{\star}\cdot\boldsymbol{\xi}]\delta[z-\boldsymbol{B}\cdot\boldsymbol{\xi}](\boldsymbol{\xi}\cdot\boldsymbol{\xi}')\delta[x'-\boldsymbol{J}\cdot\boldsymbol{\xi}']\delta[y'-\boldsymbol{B}^{\star}\cdot\boldsymbol{\xi}']\delta[y'-\boldsymbol{B}\cdot\boldsymbol{\xi}'])_{\tilde{D}}\rangle_{\tilde{D}}\rangle_{\mathrm{QRP};t}$$

It involves a conditional average of the form $\langle K[\boldsymbol{J}]\rangle_{\mathrm{QRP};t}=\int d\boldsymbol{J}\, p_t(\boldsymbol{J}|Q,R,P)K[\boldsymbol{J}]$, with

$$p_t(\boldsymbol{J}|Q,R,P) = \frac{p_t(\boldsymbol{J})\,\delta[Q-Q[\boldsymbol{J}]]\delta[R-R[\boldsymbol{J}]]\prod_{xyz}\delta[P[x,y,z]-P[x,y,z;\boldsymbol{J}]]}{\int d\boldsymbol{J}\, p_t(\boldsymbol{J})\,\delta[Q-Q[\boldsymbol{J}]]\delta[R-R[\boldsymbol{J}]]\prod_{xyz}\delta[P[x,y,z]-P[x,y,z;\boldsymbol{J}]]}$$

in which $p_t(\boldsymbol{J})$ is the weight probability density at time $t$. The solution of (6,7,8) can be used to generate the $N\to\infty$ performance measures (4) at any time:

$$E_t = \int dxdydz\, P[x,y,z]\theta[-xz] \qquad E_{\mathrm{g}} = \pi^{-1}\arccos[R/\sqrt{Q}] \qquad (9)$$

Expansion of these equations in powers of $\eta$, and retaining only the terms linear in $\eta$, gives the corresponding equations describing batch learning. So far this analysis is exact.

## 4 Closure of Macroscopic Laws

As in [6, 5] we close our macroscopic laws (6,7,8) by making the two key assumptions underlying dynamical replica theory:

(*i*)  For $N\to\infty$ our macroscopic observables obey *closed* dynamic equations.

(*ii*)  These equations are self-averaging with respect to the specific realization of $\tilde{D}$.

(*i*) implies that probability variations within $\{Q,R,P\}$ subshells are either absent or irrelevant to the macroscopic laws. We may thus make the simplest choice for $p_t(\boldsymbol{J}|Q,R,P)$:

$$p_t(\boldsymbol{J}|Q,R,P) \;\to\; \delta[Q-Q[\boldsymbol{J}]]\,\delta[R-R[\boldsymbol{J}]]\prod_{xyz}\delta[P[x,y,z]-P[x,y,z;\boldsymbol{J}]] \qquad (10)$$

The procedure (10) leads to exact laws if our observables $\{Q,R,P\}$ indeed obey closed equations for $N\to\infty$. It is a maximum entropy approximation if not. (*ii*) allows us to average the macroscopic laws over all training sets; it is observed in simulations, and proven using the formalism of [4]. Our assumptions (10) result in the closure of (6,7,8), since now the Green's function can be written in terms of $\{Q,R,P\}$. The final ingredient of dynamical replica theory is doing the average of fractions with the replica identity

$$\left\langle\frac{\int d\boldsymbol{J}\, W[\boldsymbol{J}|\tilde{D}]G[\boldsymbol{J}|\tilde{D}]}{\int d\boldsymbol{J}\, W[\boldsymbol{J}|\tilde{D}]}\right\rangle_{\mathrm{sets}} = \lim_{n\to 0}\int d\boldsymbol{J}^1\cdots d\boldsymbol{J}^n\,\langle G[\boldsymbol{J}^1|\tilde{D}]\prod_{\alpha=1}^{n}W[\boldsymbol{J}^\alpha|\tilde{D}]\rangle_{\mathrm{sets}}$$

Our problem has been reduced to calculating (non-trivial) integrals and averages. One finds that $P[x,y,z]=P[x,z|y]P[y]$ with $P[y]=(2\pi)^{-\frac{1}{2}}\exp[-\frac{1}{2}y^2]$. With the short-hands $Dy=P[y]dy$ and $\langle f(x,y,z)\rangle=\int Dydxdz\, P[x,z|y]f(x,y,z)$ we can write the resulting macroscopic laws, for the case of output noise (1), in the following compact way:

$$\frac{d}{dt}Q = 2\eta(V-\gamma Q)+\eta^2 Z \qquad\qquad \frac{d}{dt}R = \eta(W-\gamma R) \qquad (11)$$

$$\frac{\partial}{\partial t}P[x,z|y] = \frac{1}{\alpha}\int dx'\, P[x',z|y]\left\{\delta[x-x'-\eta G[x',z]]-\delta[x-x']\right\}+\frac{1}{2}\eta^2 Z\frac{\partial^2}{\partial x^2}P[x,z|y]$$

$$-\eta\frac{\partial}{\partial x}\left\{P[x,z|y]\left[U(x-Ry)+Wy-\gamma x+[V-RW-(Q-R^2)U]\Phi[x,y,z]\right]\right\} \qquad (12)$$

with

$$U=\langle\Phi[x,y,z]\mathcal{G}[x,z]\rangle, \quad V=\langle x\mathcal{G}[x,z]\rangle, \quad W=\langle y\mathcal{G}[x,z]\rangle, \quad Z=\langle\mathcal{G}^2[x,z]\rangle$$

The solution of (12) is at any time of the following form:

$$P[x,z|y] = (1-\lambda)\delta[y-z]P^+[x|y]+\lambda\delta[y+z]P^-[x|y] \qquad (13)$$

Finding the function $\Phi[x, y, z]$ (in replica symmetric ansatz) requires solving a saddle-point problem for a scalar observable $q$ and two functions $M^{\pm}[x|y]$. Upon introducing

$$B = \frac{\sqrt{qQ - R^2}}{Q(1-q)} \qquad \langle f[x,y]\rangle_{\star}^{\pm} = \frac{\int dx\, M^{\pm}[x|y]e^{Bxs}f[x,y]}{\int dx\, M^{\pm}[x|y]e^{Bxs}}$$

(with $\int dx\, M^{\pm}[x|y] = 1$ for all $y$) the saddle-point equations acquire the form

$$\text{for all } X, y: \qquad P^{\pm}[X|y] = \int Ds\, \langle \delta[X-x]\rangle_{\star}^{\pm} \tag{14}$$

$$\langle (x-Ry)^2\rangle + (qQ-R^2)[1-\frac{1}{\alpha}] = \frac{qQ+Q-2R^2}{\sqrt{qQ-R^2}} \int DyDs\, s[(1-\lambda)\langle x\rangle_{\star}^{+} + \lambda\langle x\rangle_{\star}^{-}] \tag{15}$$

The equations (14) which determine $M^{\pm}[x|y]$ have the same structure as the corresponding (single) equation in [5, 6], so the proofs in [5, 6] again apply, and the solutions $M^{\pm}[x|y]$, given a $q$ in the physical range $q \in [R^2/Q, 1]$, are unique. The function $\Phi[x, y, z]$ is then given by

$$\Phi[X,y,z] = \int \frac{Ds\, s}{\sqrt{qQ-R^2}P[X,z|y]} \left\{ (1-\lambda)\delta[z-y]\langle\delta[X-x]\rangle_{\star}^{+} + \lambda\delta[z+y]\langle\delta[X-x]\rangle_{\star}^{-} \right\} \tag{16}$$

Working out predictions from these equations is generally CPU-intensive, mainly due to the functional saddle-point equation (14) to be solved at each time step. However, as in [7] one can construct useful approximations of the theory, with increasing complexity:

(i) Large $\alpha$ approximation (giving the simplest theory, without saddle-point equations)
(ii) Conditionally Gaussian approximation for $M[x|y]$ (with $y$-dependent moments)
(iii) Annealed approximation of the functional saddle-point equation

## 5   Benchmark Tests: The Limits $\alpha \to \infty$ and $\lambda \to 0$

We first show that in the limit $\alpha \to \infty$ our theory reduces to the simple $(Q, R)$ formalism of infinite training sets, as worked out for noisy teachers in [12]. Upon making the ansatz

$$P^{\pm}[x|y] = P[x|y] = [2\pi(Q-R^2)]^{-\frac{1}{2}} e^{-\frac{1}{2}[x-Ry]^2/(Q-R^2)} \tag{17}$$

one finds

$$M^{\pm}[x|y] = P[x|y], \qquad q = R^2/Q, \qquad \Phi[x,y,z] = (x-Ry)/(Q-R^2)$$

Insertion of our ansatz into (12), followed by rearranging of terms and usage of the above expression for $\Phi[x, y, z]$, shows that (12) is satisfied. The remaining equations (11) involve only averages over the Gaussian distribution (17), and indeed reduce to those of [12]:

$$\frac{1}{\eta}\frac{d}{dt}Q = (1-\lambda)\left\{2\langle x\mathcal{G}[x,y]\rangle + \eta\langle\mathcal{G}^2[x,y]\rangle\right\} + \lambda\left\{2\langle x\mathcal{G}[x,-y]\rangle + \eta\langle\mathcal{G}^2[x,-y]\rangle\right\} - 2\gamma Q$$

$$\frac{1}{\eta}\frac{d}{dt}R = (1-\lambda)\langle y\mathcal{G}[x,y]\rangle + \lambda\langle y\mathcal{G}[x,-y]\rangle - \gamma R$$

Next we turn to the limit $\lambda \to 0$ (restricted training sets & noise-free teachers) and show that here our theory reproduces the formalism of [6, 5]. Now we make the following ansatz:

$$P^{+}[x|y] = P[x|y], \qquad P[x,z|y] = \delta[z-y]P[x|y] \tag{18}$$

Insertion shows that for $\lambda = 0$ solutions of this form indeed solve our equations, giving $\Phi[x, y, z] \to \Phi[x, y]$ and $M^{+}[x|y] = M[x|y]$, and leaving us exactly with the formalism of [6, 5] describing the case of noise-free teachers and restricted training sets (apart from some new terms due to the presence of weight decay, which was absent in [6, 5]).

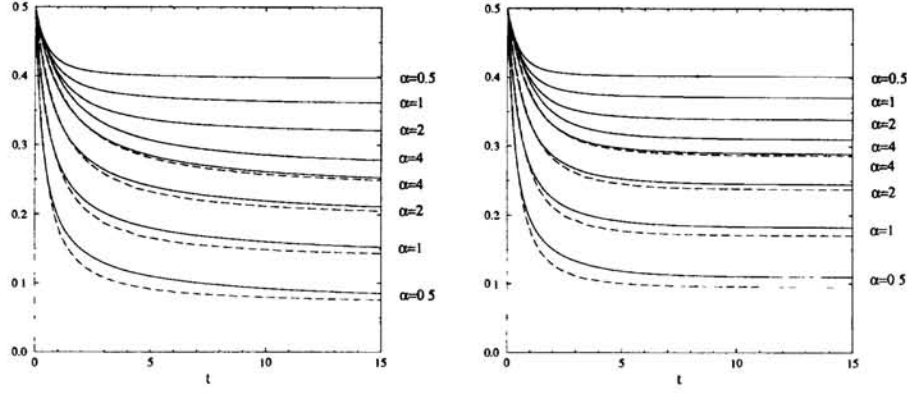

Figure 1: On-line Hebbian learning: conditionally Gaussian approximation versus exact solution in [9] ($\eta = 1$, $\lambda = 0.2$). Left: $\gamma = 0.1$, right: $\gamma = 0.5$. Solid lines: approximated theory, dashed lines: exact result. Upper curves: $E_g$ as functions of time (here the two theories agree), lower curves: $E_t$ as functions of time.

## 6  Benchmark Tests: Hebbian Learning

The special case of Hebbian learning, i.e. $\mathcal{G}[x, z] = \mathrm{sgn}(z)$, can be solved exactly at any time, for arbitrary $\{\alpha, \lambda, \gamma\}$ [9], providing yet another excellent benchmark for our theory. For batch execution of Hebbian learning the macroscopic laws are obtained upon expanding (11,12) and retaining only those terms which are linear in $\eta$. All integrations can now be done and all equations solved explicitly, resulting in $U = 0$, $Z = 1$, $W = (1-2\lambda)\sqrt{2/\pi}$, and

$$Q = Q_0\, e^{-2\eta\gamma t} + \frac{2R_0(1-2\lambda)}{\gamma} e^{-\eta\gamma t}[1 - e^{-\eta\gamma t}]\sqrt{\frac{2}{\pi}} + \left[\frac{2}{\pi}(1-2\lambda)^2 + \frac{1}{\alpha}\right]\frac{[1 - e^{-\eta\gamma t}]^2}{\gamma^2}$$

$$R = R_0\, e^{-\eta\gamma t} + (1-2\lambda)\sqrt{2/\pi}\,[1 - e^{-\eta\gamma t}]/\gamma \qquad q = [\alpha R^2 + (1-e^{-\eta\gamma t})^2/\gamma^2]/\alpha Q$$

$$P^{\pm}[x|y] = \left[2\pi(Q - R^2)\right]^{-\frac{1}{2}} e^{-\frac{1}{2}[x - Ry \mp\, \mathrm{sgn}(y)[1-e^{-\eta\gamma t}]/\alpha\gamma]^2/(Q-R^2)} \tag{19}$$

From these results, in turn, follow the performance measures $E_g = \pi^{-1}\arccos[R/\sqrt{Q}]$ and

$$E_t = \frac{1}{2} - \frac{1}{2}(1-\lambda)\int Dy\,\mathrm{erf}\left[\frac{|y|R + [1-e^{-\eta\gamma t}]/\alpha\gamma}{\sqrt{2(Q-R^2)}}\right] + \frac{1}{2}\lambda\int Dy\,\mathrm{erf}\left[\frac{|y|R - [1-e^{-\eta\gamma t}]/\alpha\gamma}{\sqrt{2(Q-R^2)}}\right]$$

Comparison with the exact solution, calculated along the lines of [9] or, equivalently, obtained upon putting $t \ll \eta^{-2}$ in [9], shows that the above expressions are all exact.

For on-line execution we cannot (yet) solve the functional saddle-point equation in general. However, some analytical predictions can still be extracted from (11,12,13):

$$Q = Q_0\, e^{-2\eta\gamma t} + \frac{2R_0(1-2\lambda)}{\gamma} e^{-\eta\gamma t}[1 - e^{-\eta\gamma t}]\sqrt{\frac{2}{\pi}} + \left[\frac{2}{\pi}(1-2\lambda)^2 + \frac{1}{\alpha}\right]\frac{[1 - e^{-\eta\gamma t}]^2}{\gamma^2}$$

$$R = R_0\, e^{-\eta\gamma t} + (1-2\lambda)\sqrt{2/\pi}\,[1 - e^{-\eta\gamma t}]/\gamma \qquad\qquad + \frac{\eta}{2\gamma}[1 - e^{-2\eta\gamma t}]$$

$$\int dx\, x P^{\pm}[x|y] = Ry \pm\, \mathrm{sgn}(y)[1 - e^{-\eta\gamma t}]/\alpha\gamma$$

with $U = 0$, $W = (1-2\lambda)\sqrt{2/\pi}$, $V = WR + [1-e^{-\eta\gamma t}]/\alpha\gamma$, and $Z = 1$. Comparison with the results in [9] shows that the above expressions, and thus also that of $E_g$, are all fully exact, at any time. Observables involving $P[x, y, z]$ (including the training error) are not as easily solved from our equations. Instead we used the conditionally Gaussian approximation (found to be adequate for the noiseless Hebbian case [5, 6, 7]). The result is shown in figure 1. The agreement is reasonable, but significantly less than that in [6]; apparently teacher noise adds to the deformation of the field distribution away from a Gaussian shape.

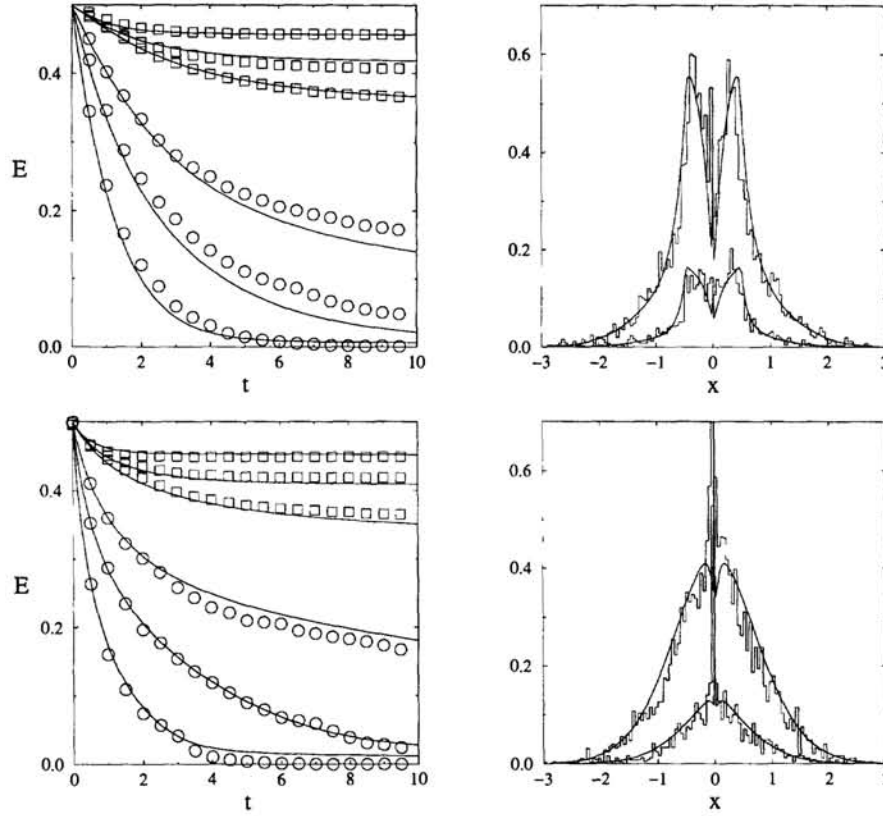

Figure 2: Large $\alpha$ approximation versus numerical simulations (with $N = 10,000$), for $\gamma = 0$ and $\lambda = 0.2$. Top row: Perceptron rule, with $\eta = \frac{1}{2}$. Bottom row: Adatron rule, with $\eta = \frac{3}{2}$. Left: training errors $E_t$ and generalisation errors $E_g$ as functions of time, for $\alpha \in \{\frac{1}{2}, 1, 2\}$. Lines: approximated theory, markers: simulations (circles: $E_t$, squares: $E_g$). Right: joint distributions for student field and teacher noise $P^\pm[x] = \int dy \, P[x, y, z = \pm y]$ (upper: $P^+[x]$, lower: $P^-[x]$). Histograms: simulations, lines: approximated theory.

## 7   Non-Linear Learning Rules: Theory versus Simulations

In the case of non-linear learning rules no exact solution is known against which to test our formalism, leaving numerical simulations as the yardstick. We have evaluated numerically the large $\alpha$ approximation of our theory for Perceptron learning, $\mathcal{G}[x, z] = \operatorname{sgn}(z)\theta[-xz]$, and for Adatron learning, $\mathcal{G}[x, z] = \operatorname{sgn}(z)|z|\theta[-xz]$. This approximation leads to the following fully explicit equation for the field distributions:

$$\frac{d}{dt} P^\pm[x|y] = \frac{1}{\alpha} \int dx' P^\pm[x'|y] \left\{ \delta[x - x' - \eta\mathcal{F}[x', \pm y]] - \delta[x - x'] \right\} + \frac{1}{2}\eta^2 Z \frac{\partial^2}{\partial x^2} P^\pm[x|y]$$

with
$$- \eta \frac{\partial}{\partial x} \left\{ P[x|y] \left[ Wy - \gamma x + \frac{U[\overline{x}^\pm(y) - Ry] + (V - RW)[x - \overline{x}^\pm(y)]}{Q - R^2} \right] \right\}$$

$$U = \int Dy dx \left\{ (1 - \lambda)P^+[x|y][x - \overline{x}^+(y)]\mathcal{G}[x, y] + \lambda P^-[x|y][x - \overline{x}^-(y)]\mathcal{G}[x, -y] \right\}$$

$$V = \int Dy dx \, x \left\{ (1 - \lambda)P^+[x|y]\mathcal{G}[x, y] + \lambda P^-[x|y]\mathcal{G}[x, -y] \right\}$$

$$W = \int Dy dx \, y \left\{ (1 - \lambda)P^+[x|y]\mathcal{G}[x, y] + \lambda P^-[x|y]\mathcal{G}[x, -y] \right\}$$

$$Z = \int Dy dx \left\{ (1 - \lambda)P^+[x|y]\mathcal{G}^2[x, y] + \lambda P^-[x|y]\mathcal{G}^2[x, -y] \right\}$$

and with the short-hands $\overline{x}^{\pm}(y) = \int dx\, xP^{\pm}[x|y]$. The result of our comparison is shown in figure 2. Note: $E_t$ increases monotonically with $\alpha$, and $E_g$ decreases monotonically with $\alpha$, at any $t$. As in the noise-free formalism [7], the large $\alpha$ approximation appears to capture the dominant terms both for $\alpha \to \infty$ and for $\alpha \to 0$. The predicting power of our theory is mainly limited by numerical constraints. For instance, the Adatron learning rule generates singularities at $x = 0$ in the distributions $P^{\pm}[x|y]$ (especially for small $\eta$) which, although predicted by our theory, are almost impossible to capture in numerical solutions.

## 8   Discussion

We have shown how a recent theory to describe the dynamics of supervised learning with restricted training sets (designed to apply in the data recycling regime, and for arbitrary on-line and batch learning rules) [5, 6, 7] in large layered neural networks can be generalized successfully in order to deal also with noisy teachers. In our generalized approach the joint distribution $P[x, y, z]$ for the fields of student, 'clean' teacher, and noisy teacher is taken to be a dynamical order parameter, in addition to the conventional observables $Q$ and $R$. From the order parameter set $\{Q, R, P\}$ we derive the generalization error $E_g$ and the training error $E_t$. Following the prescriptions of dynamical replica theory one finds a diffusion equation for $P[x, y, z]$, which we have evaluated by making the replica-symmetric ansatz. We have carried out several orthogonal benchmark tests of our theory: ($i$) for $\alpha \to \infty$ (no data recycling) our theory is exact, ($ii$) for $\lambda \to 0$ (no teacher noise) our theory reduces to that of [5, 6, 7], and ($iii$) for batch Hebbian learning our theory is exact. For on-line Hebbian learning our theory is exact with regard to the predictions for $Q$, $R$, $E_g$ and the $y$-dependent conditional averages $\int dx\, xP^{\pm}[x|y]$, at any time, and a crude approximation of our equations already gives reasonable agreement with the exact results [9] for $E_t$. For non-linear learning rules (Perceptron and Adatron) we have compared numerical solution of a simple large $\alpha$ aproximation of our equations to numerical simulations, and found satisfactory agreement. This paper is a preliminary presentation of results obtained in the second stage of a research programme aimed at extending our theoretical tools in the arena of learning dynamics, building on [5, 6, 7]. Ongoing work is aimed at systematic application of our theory and its approximations to various types of non-linear learning rules, and at generalization of the theory to multi-layer networks.

## References

[1] Mace C.W.H. and Coolen A.C.C (1998), *Statistics and Computing* **8**, 55

[2] Saad D. (ed.) (1998), *On-Line Learning in Neural Networks* (Cambridge: CUP)

[3] Hertz J.A., Krogh A. and Thorgersson G.I. (1989), *J. Phys. A* **22**, 2133

[4] Horner H. (1992a), *Z. Phys. B* **86**, 291 *and* Horner H. (1992b), *Z. Phys. B* **87**, 371

[5] Coolen A.C.C. and Saad D. (1998), in *On-Line Learning in Neural Networks*, Saad D. (ed.), (Cambridge: CUP)

[6] Coolen A.C.C. and Saad D. (1999), in Advances in Neural Information Processing Systems 11, Kearns D., Solla S.A., Cohn D.A. (eds.), (MIT press)

[7] Coolen A.C.C. and Saad D. (1999), *preprints KCL-MTH-99-32 & KCL-MTH-99-33*

[8] Rae H.C., Sollich P. and Coolen A.C.C. (1999), in Advances in Neural Information Processing Systems 11, Kearns D., Solla S.A., Cohn D.A. (eds.), (MIT press)

[9] Rae H.C., Sollich P. and Coolen A.C.C. (1999), *J. Phys. A* **32**, 3321

[10] Inoue J.I. (1999) *private communication*

[11] Wong K.Y.M., Li S. and Tong Y.W. (1999), *preprint cond-mat/9909004*

[12] Biehl M., Riegler P. and Stechert M. (1995), *Phys. Rev. E* **52**, 4624
